# Rate-coded Restricted Boltzmann Machines for Face Recognition

**Yee Whye Teh**
Department of Computer Science
University of Toronto
Toronto M5S 2Z9 Canada
*ywteh@cs.toronto.edu*

**Geoffrey E. Hinton**
Gatsby Computational Neuroscience Unit*
University College London
London WC1N 3AR U.K.
*hinton@gatsby.ucl.ac.uk*

## Abstract

We describe a neurally-inspired, unsupervised learning algorithm that builds a non-linear generative model for pairs of face images from the same individual. Individuals are then recognized by finding the highest relative probability pair among all pairs that consist of a test image and an image whose identity is known. Our method compares favorably with other methods in the literature. The generative model consists of a single layer of rate-coded, non-linear feature detectors and it has the property that, given a data vector, the true posterior probability distribution over the feature detector activities can be inferred rapidly without iteration or approximation. The weights of the feature detectors are learned by comparing the correlations of pixel intensities and feature activations in two phases: When the network is observing real data and when it is observing reconstructions of real data generated from the feature activations.

## 1 Introduction

Face recognition is difficult when the number of individuals is large and the test and training images of an individual differ in expression, pose, lighting or the date on which they were taken. In addition to being an important application, face recognition allows us to evaluate different kinds of algorithm for learning to recognize or compare objects, since it requires accurate representation of fine discriminative features in the presence of relatively large within-individual variations. This is made even more difficult when there are very few exemplars of each individual.

We start by describing a new unsupervised learning algorithm for a restricted form of Boltzmann machine [1]. We then show how to generalize the generative model and the learning algorithm to deal with real-valued pixel intensities and rate-coded feature detectors. We then apply the model to face recognition and compare it to other methods.

## 2 Inference and learning in Restricted Boltzmann Machines

A Restricted Boltzmann machine (RBM) [2] is a Boltzmann machine with a layer of visible units and a single layer of hidden units with no hidden-to-hidden nor visible-to-visible

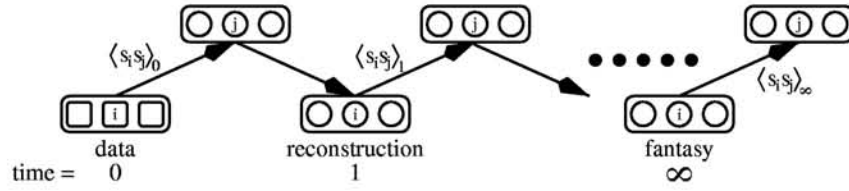

Figure 1: Alternating Gibbs sampling and the terms in the learning rules of a RBM.

connections. Because there is no explaining away [3], inference in an RBM is much easier than in a general Boltzmann machine or in a causal belief network with one hidden layer. There is no need to perform any iteration to determine the activities of the hidden units, as the hidden states, $s_j$, are *conditionally independent* given the visible states, $s_i$. The distribution of $s_j$ is given by the standard logistic function:

$$p(s_j = 1|s_i) = \frac{1}{1 + \exp(-\sum_i w_{ij} s_i)} \qquad (1)$$

Conversely, the hidden states of an RBM are *marginally dependent* so it is easy for an RBM to learn population codes in which units may be highly correlated. It is hard to do this in causal belief networks with one hidden layer because the generative model of a causal belief net assumes marginal independence.

An RBM can be trained using the standard Boltzmann machine learning algorithm which follows a noisy but unbiased estimate of the gradient of the log likelihood of the data. One way to implement this algorithm is to start the network with a data vector on the visible units and then to alternate between updating all of the hidden units in parallel and updating all of the visible units in parallel with Gibbs sampling. Figure 1 illustrates this process. If this alternating Gibbs sampling is run to equilibrium, there is a very simple way to update the weights so as to minimize the Kullback-Leibler divergence, $Q^0 \| Q^\infty$, between the data distribution, $Q^0$, and the equilibrium distribution of fantasies over the visible units, $Q^\infty$, produced by the RBM [4]:

$$\Delta w_{ij} \propto <s_i s_j>_{Q^0} - <s_i s_j>_{Q^\infty} \qquad (2)$$

where $<s_i s_j>_{Q^0}$ is the expected value of $s_i s_j$ when data is clamped on the visible units and the hidden states are sampled from their conditional distribution given the data, and $<s_i s_j>_{Q^\infty}$ is the expected value of $s_i s_j$ after prolonged Gibbs sampling.

This learning rule does not work well because it can take a long time to approach equilibrium and the sampling noise in the estimate of $<s_i s_j>_{Q^\infty}$ can swamp the gradient. Hinton [1] shows that it is far more effective to minimize the *difference* between $Q^0 \| Q^\infty$ and $Q^1 \| Q^\infty$ where $Q^1$ is the distribution of the one-step reconstructions of the data that are produced by first picking binary hidden states from their conditional distribution given the data and then picking binary visible states from their conditional distribution given the hidden states. The exact gradient of this "contrastive divergence" is complicated because the distribution $Q^1$ depends on the weights, but this dependence can safely be ignored to yield a simple and effective learning rule for following the approximate gradient of the contrastive divergence:

$$\Delta w_{ij} \propto <s_i s_j>_{Q^0} - <s_i s_j>_{Q^1} \qquad (3)$$

## 3 Applying RBMs to face recognition

For images of faces, binary pixels are far from ideal. A simple way to increase the representational power without changing the inference and learning procedures is to imagine that

each visible unit, $i$, has 10 replicas which all have identical weights to the hidden units. So far as the hidden units are concerned, it makes no difference which particular replicas are turned on: it is only the number of active replicas that counts. So a pixel can now have 11 different intensities. During reconstruction of the image from the hidden activities, all the replicas can share the computation of the probability, $p_i$, of turning on, and then we can select $n$ replicas to be on with probability $\binom{10}{n} n^{p_i} (10-n)^{(1-p_i)}$. We actually approximated this binomial distribution by just adding a little Gaussian noise to $10p_i$ and rounding. The same trick can be used for the hidden units. Eq. 3 is unaffected except that $s_i$ and $s_j$ are now the number of active replicas.

The replica trick can be seen as a way of simulating a single neuron over a time interval in which it may produce multiple spikes that constitute a rate-code. For this reason we call the model "RBMrate". We assumed that the visible units can produce up to 10 spikes and the hidden units can produce up to 100 spikes. We also made two further approximations: We replaced $s_i$ and $s_j$ in Eq. 3 by their expected values and we used the expected value of $s_i$ when computing the probability of activation of the hidden units. However, we continued to use the stochastically chosen integer firing rates of the hidden units when computing the one-step reconstructions of the data, so the hidden activities cannot transmit an unbounded amount of information from the data to the reconstruction.

A simple way to use RBMrate for face recognition is to train a single model on the training set, and to identify a face by finding the gallery image that produces a hidden activity vector that is most similar to the one produced by the face. This is how eigenfaces are used for recognition, but it does not work well because it does not take into account the fact that some variations across faces are important for recognition, while some variations are not. To correct this, we instead trained an RBMrate model on pairs of different images of the same individual, and then we used this model of pairs to decide which gallery image is best paired with the test image. To account for the fact that the model likes some individual face images more than others, we define the fit between two faces $f_1$ and $f_2$ as $G(f_1, f_2) + G(f_2, f_1) - G(f_1, f_1) - G(f_2, f_2)$ where the goodness score $G(v_1, v_2)$ is the negative free energy of the image pair $v_1, v_2$ under the model. Weight-sharing is not used, hence $G(v_1, v_2) \neq G(v_2, v_1)$. However, to preserve symmetry, each pair of images of the same individual $v_1, v_2$ in the training set has a reversed pair $v_2, v_1$ in the set. We trained the model with 100 hidden units on 1000 image pairs (500 distinct pairs) for 2000 iterations in batches of 100, with a learning rate of $2.5 \times 10^{-6}$ for the weights, a learning rate of $5 \times 10^{-6}$ for the biases, and a momentum of 0.95.

One advantage of eigenfaces over correlation is that once the test image has been converted into a vector of eigenface activations, comparisons of test and gallery images can be made in the low-dimensional space of eigenface activations rather than the high-dimensional space of pixel intensities. The same applies to our face-pair network, as the goodness score of an image pair is a simple function of the total input received by each hidden unit from each image. The total inputs from each gallery image can be precomputed and stored, while the total inputs from a test image only needs to be computed once for comparisons with all gallery images.

## 4 The FERET database

Our version of the FERET database contained 1002 frontal face images of 429 individuals taken over a period of a few years under varying lighting conditions. Of these images, 818 are used as both the gallery and the training set and the remaining 184 are divided into four disjoint test sets:

The **Δexpression** test set contains 110 images of different individuals. These individuals all have another image in the training set that was taken with the same lighting conditions

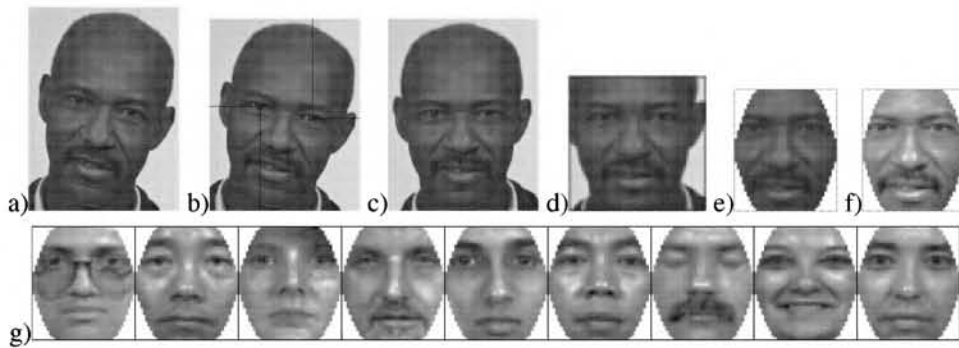

Figure 2: Images are normalized in five stages: a) Original image; b) Locate centers of eyes by hand; c) Rotate image; d) Crop image and subsample at $56 \times 56$ pixels; e) Mask out all of the background and some of the face, leaving 1768 pixels in an oval shape; f) Equalize the intensity histogram; g) Some examples of processed images.

at the same time but with a different expression. The training set also includes a further 244 pairs of images that differ only in expression.

The $\Delta$**days** test set contains 40 images that come from 20 individuals. Each of these individuals has two images from the same session in the training set and two images taken in a session 4 days later or earlier in the test set. A further 28 individuals were photographed 4 days apart and all 112 of these images are in the training set.

The $\Delta$**months** test set is just like the $\Delta$days test set except that the time between sessions was at least three months and different lighting conditions were present in the two sessions. This set contains 20 images of 10 individuals. A further 36 images of 9 more individuals were included in the training set.

The $\Delta$**glasses** test set contains 14 images of 7 different individuals. Each of these individuals has two images in the training set that were taken in another session on the same day. The training and test pairs for an individual differ in that one pair has glasses and the other does not. The training set includes a further 24 images, half with glasses and half without, from 6 more individuals.

The images include the whole head, parts of the shoulder, and background. Instead of working with whole images, which contain much irrelevant information, we worked with face images that were normalized as shown in figure 2. Masking out all of the background inevitably looses the contour of the face which contains much discriminative information. The histogram equalization step removes most lighting effects, but it also removes some relevant information like the skin tone. For the best performance, the contour shape and skin tone would have to be used as additional sources of discriminative information.

## 5   Comparative results

We compared RBMrate with four popular face recognition methods. The first and simplest is **correlation**, which returns the similarity score as the angle between two images represented as vectors of pixel intensities. This performed better than using the Euclidean distance as a score. The second method is **eigenfaces** [5], which first projects the images onto the principal component subspaces, then returns the similarity score as the angle between the projected images. The third method is **fisherfaces** [6]. Instead of projecting the images onto the subspace of the principal components, which maximizes the variance

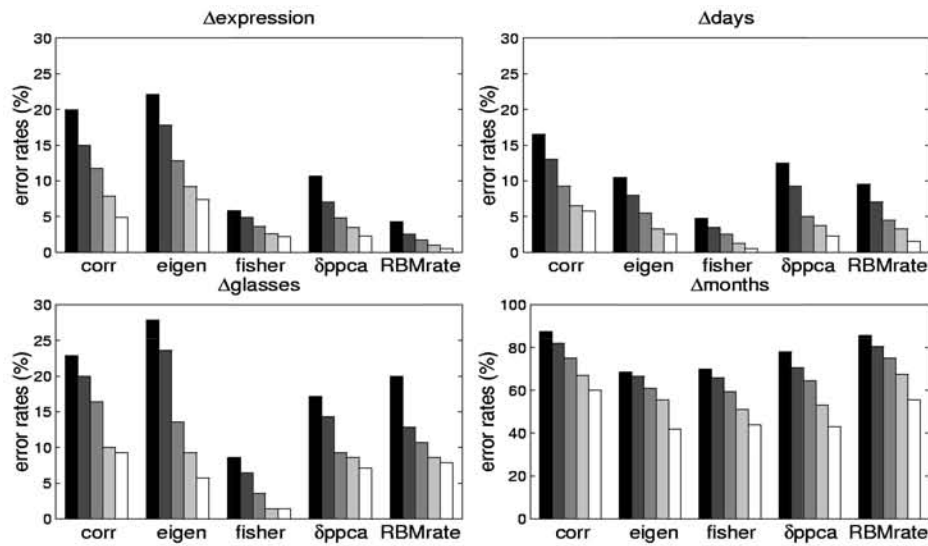

Figure 3: Error rates of all methods on all test sets. The bars in each group correspond, from left to right, to the rank-1, rank-2, rank-4, rank-8 and rank-16 error rates. The rank-$n$ error rate is the percentage of test images where the $n$ most similar gallery images are all incorrect.

among the projected images, fisherfaces projects the images onto a subspace which, at the same time, maximizes the between individual variances and minimizes the within individual variances in the training set. The final method, which we shall call **$\delta$ppca**, is proposed by Moghaddam *et al* [7]. This method models differences between images of the same individual as a PPCA [8, 9], and differences between images of different individuals as another PPCA. Then given a difference of two images, it returns as the similarity score the likelihood ratio of the difference image under the two PPCA models. It was the best performing algorithm in the September 1996 FERET test [10].

For eigenfaces, we used 199 principal components, omitting the first principal component, as we determined manually that it encodes simply for lighting conditions. This improved the recognition performances on all the test sets except for $\Delta$expression . We used a subspace of dimension 200 for fisherfaces, while we used 10 and 30 dimensional PPCAs for the within-class and between-class model of $\delta$ppca respectively. These are the same numbers used by Moghaddam *et al* and gives the best results in our simulations. The number of dimensions or hidden units used by each method was optimized for that particular method for best performance.

Figure 3 shows the error rates of all five methods on the test sets. The results were averaged over 10 random partitions of the dataset to improve statistical significance. Correlation and eigenfaces perform poorly on $\Delta$expression , probably because they do not attempt to ignore the within-individual variations, whereas the other methods do. All the models did very poorly on the $\Delta$months test set which is unfortunate as this is the test set that is most like real applications. RBMrate performed best on $\Delta$expression , fisherfaces is best on $\Delta$days and $\Delta$glasses , while eigenfaces is best on $\Delta$months . These results show that RBMrate is competitive with but do not perform better than other methods. Figure 4 shows that after our preprocessing, human observers also have great difficulty with the $\Delta$months test set, probably because the task is intrinsically difficult and is made even harder by the loss of contour and skin tone information combined with the misleading oval

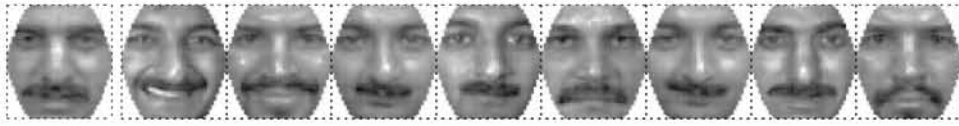

Figure 4: On the left is a test image from Δmonths and on the right are the 8 most similar images returned by RBMrate . Most human observers cannot find the correct match within these 8.

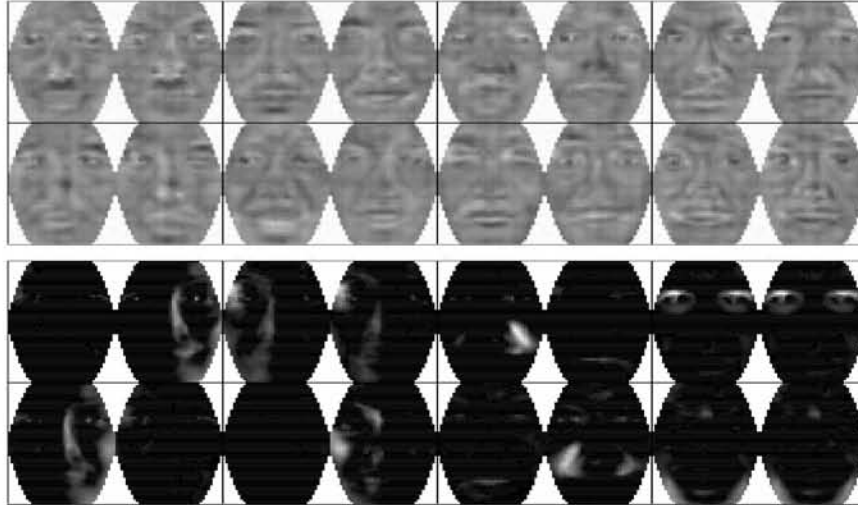

Figure 5: Example features learned by RBMrate . Each pair of RFs constitutes a feature. Top half: with unconstrained weights; bottom half: with non-negative weight constraints.

contour produced by masking out all of the background.

## 6 Receptive fields learned by RBMrate

The top half of figure 5 shows the weights of a few of the hidden units after training. All the units encode global features, probably because the image normalization ensures that there are strong long range correlations in pixel intensities. The maximum size of the weights is 0.01765, with most weights having magnitudes smaller than 0.005. Note, however, that the hidden unit activations range from 0 to 100.

On the left are 4 units exhibiting interesting features and on the right are 4 units chosen at random. The top unit of the first column seems to be encoding the presence of mustache in both faces. The bottom unit seems to be coding for prominent right eyebrows in both faces. Note that these are facial features which often remain constant across images of the same individual. In the second column are two features which seem to encode for different facial expressions in the two faces. The right side of the top unit encodes a smile while the left side is expressionless. This is reversed in the bottom unit. So the network has discovered some features which are fairly constant across images in the same class, and some features which can differ substantially within a class.

Inspired by [11], we tried to enforce local features by restricting the weights to be non-

negative. This is achieved by resetting negative weights to zero after each weight update. The bottom half of figure 5 shows some of the hidden receptive fields learned. Except for the 4 features on the left, all other features are local and code for features like mouth shape changes (third column) and eyes and cheeks (fourth column). The 4 features on the left are much more global and clearly capture the fact that the direction of the lighting can differ for two images of the same person. Unfortunately, constraining the weights to be non-negative strongly limits the representational power of RBMrate and makes it worse than all the other methods on all the test sets.

## 7 Conclusions

We have introduced a new method for face recognition based on a non-linear generative model. The generative model can be very complex, yet retains the efficiency required for applications. Performance on the FERET database is comparable to popular methods. However, unlike other methods based on linear models, there is plenty of room for further development using prior knowledge to constrain the weights or additional layers of hidden units to model the correlations of feature detector activities. These improvements should translate into improvements in the rate of recognition.

### Acknowledgements

We thank Jonathon Phillips for graciously providing us with the FERET database, the referees for useful comments and the Gatsby Charitable Foundation for funding.

## Footnotes

*Correspondence address

### References

[1] G. E. Hinton. Training products of experts by minimizing contrastive divergence. Technical Report GCNU TR 2000-004, Gatsby Computational Neuroscience Unit, University College London, 2000.

[2] P. Smolensky. Information processing in dynamical systems: Foundations of harmony theory. In D. E. Rumelhart and J. L. McClelland, editors, *Parallel Distributed Processing: Explorations in the Microstructure of Cognition. Volume 1: Foundations*. MIT Press, 1986.

[3] J. Pearl. *Probabilistic reasoning in intelligent systems : networks of plausible inference*. Morgan Kaufmann Publishers, San Mateo CA, 1988.

[4] G. E. Hinton and T. J. Sejnowski. Learning and relearning in boltzmann machines. In D. E. Rumelhart and J. L. McClelland, editors, *Parallel Distributed Processing: Explorations in the Microstructure of Cognition. Volume 1: Foundations*. MIT Press, 1986.

[5] M. Turk and A. Pentland. Eigenfaces for recognition. *Journal of Cognitive Neuroscience*, 3(1):71–86, 1991.

[6] P. N. Belmumeur, J. P. Hespanha, and D. J. Kriegman. Eigenfaces versus fisherfaces: recognition using class specific linear projection. In *European Conference on Computer Vision*, 1996.

[7] B. Moghaddam, W. Wahid, and A. Pentland. Beyond eigenfaces: probabilistic matching for face recognition. In *IEEE International Conference on Automatic Face and Gesture Recognition*, 1998.

[8] B. Moghaddam and A. Pentland. Probabilistic visual learning for object representation. *IEEE Transactions on Pattern Analysis and Machine Intelligence*, 19(7):696–710, 1997.

[9] M. E. Tipping and C. M. Bishop. Probabilistic principal component analysis. Technical Report NCRG/97/010, Neural Computing Research Group, Aston University, 1997.

[10] P. J. Phillips, H. Moon, P. Rauss, and S. A. Rizvi. The FERET september 1996 database and evaluation procedure. In *International Conference on Audio and Video-based Biometric Person Authentication*, 1997.

[11] D. Lee and H. S. Seung. Learning the parts of objects by non-negative matrix factorization. *Nature*, 401, October 1999.
